# Backpropagation without Multiplication

**Patrice Y. Simard**
AT&T Bell Laboratories
Holmdel, NJ 07733

**Hans Peter Graf**
AT&T Bell Laboratories
Holmdel, NJ 07733

## Abstract

The back propagation algorithm has been modified to work without any multiplications and to tolerate computations with a low resolution, which makes it more attractive for a hardware implementation. Numbers are represented in floating point format with 1 bit mantissa and 3 bits in the exponent for the states, and 1 bit mantissa and 5 bit exponent for the gradients, while the weights are 16 bit fixed-point numbers. In this way, all the computations can be executed with shift and add operations. Large networks with over 100,000 weights were trained and demonstrated the same performance as networks computed with full precision. An estimate of a circuit implementation shows that a large network can be placed on a single chip, reaching more than 1 billion weight updates per second. A speedup is also obtained on any machine where a multiplication is slower than a shift operation.

## 1 INTRODUCTION

One of the main problems for implementing the backpropagation algorithm in hardware is the large number of multiplications that have to be executed. Fast multipliers for operands with a high resolution require a large area. Hence the multipliers are the elements dominating the area of a circuit. Many researchers have tried to reduce the size of a circuit by limiting the resolution of the computation. Typically, this is done by simply reducing the number of bits utilized for the computation. For a forward pass a reduction to just a few, 4 to 6, bits, often degrades the performance very little, but learning requires considerably more resolution. Requirements ranging anywhere from 8 bits to more than 16 bits were reported to be necessary to make learning converge reliably (Sakaue et al., 1993; Asanovic, Morgan and Wawrzynek, 1993; Reyneri and Filippi, 1991). But there is no general theory, how much resolution is enough, and it depends on several factors, such as the size and architecture of the network as well as on the type of problem to be solved.

Several researchers have tried to train networks where the weights are limited to powers of two (Kwan and Tang, 1993; White and Elmasry, 1992; Marchesi et al., 1993). In this way all the multiplications can be reduced to shift operations, an operation that can be implemented with much less hardware than a multiplication. But restricting the weight values severely impacts the performance of a network, and it is tricky to make the learning procedure converge. In fact, some researchers keep weights with a full resolution off-line and update these weights in the backward pass, while the weights with reduced resolution are used in the forward pass (Marchesi et al., 1993). Similar tricks are usually used when networks implemented in analog hardware are trained. Weights with a high resolution are stored in an external, digital memory while the analog network with its limited resolution is used in the forward pass. If a high resolution copy is not stored, the weight update process needs to be modified. This is typically done by using a stochastic update technique, such as weight dithering (Vincent and Myers, 1992), or weight perturbation (Jabri and Flower, 1992).

We present here an algorithm that instead of reducing the resolution of the weights, reduces the resolution of all the other values, namely those of the states, gradients and learning rates, to powers of two. This eliminates multiplications without affecting the learning capabilities of the network. Therefore we obtain the benefit of a much compacter circuit without any compromises on the learning performance. Simulations of large networks with over 100,000 weights show that this algorithm performs as well as standard backpropagation computed with 32 bit floating point precision.

## 2  THE ALGORITHM

The forward propagation for each unit $i$, is given by the equation:

$$x_j = f_j(\sum_i w_{ji} x_i) \tag{1}$$

where $f$ is the unit function, $w_{ji}$ is the weight from unit $i$ to unit $j$, and $x_i$ is the activation of unit $i$. The backpropagation algorithm is robust with regard to the unit function as long as the function is nonlinear, monotonically increasing, and a derivative exists (the most commonly used function is depicted in Figure 1, left. A saturated ramp function (see Figure 1, middle), for instance, performs as well as the differentiable sigmoid. The binary threshold function, however, is too much of a simplification and results in poor performance. The choice of our function is dictated by the fact that we would like to have only powers of two for the unit values. This function is depicted in Figure 1, right. It gives performances comparable to the sigmoid or the saturated ramp. Its values can be represented by a 1 bit mantissa (the sign) with a 2 or 3 bit exponent (negative powers of two).

The derivative of this function is a sum of Dirac delta functions, but we take instead the derivative of a piecewise linear ramp function (see Figure 1). One could actually consider this a low pass filtered version of the real derivative. After the gradients of all the units have been computed using the equation,

$$g_i = f_i'(sum_i) \sum_j w_{ji} g_j \tag{2}$$

we will discretize the values to be a power of two (with sign). This introduces noise into the gradient and its effect on the learning has to be considered carefully. This

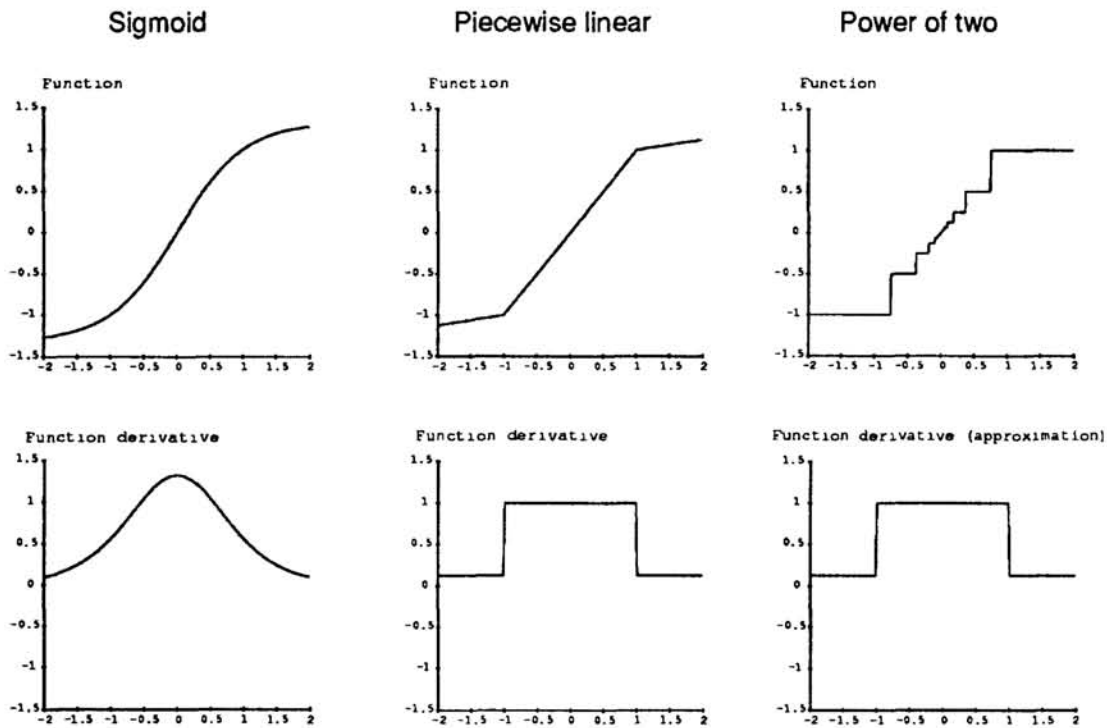

Figure 1: Left: sigmoid function with its derivative. Middle: piecewise linear function with its derivative. Right: Saturated power of two function with a power of two approximation of its derivative (identical to the piecewise linear derivative).

problem will be discussed in section 4. The backpropagation algorithm can now be implemented with additions (or subtractions) and shifting only. The weight update is given by the equation:

$$\Delta w_{ji} = -\eta g_j x_i \tag{3}$$

Since both $g_j$ and $x_i$ are powers of two, the weight update also reduces to additions and shifts.

## 3    RESULTS

A large structured network with five layers and overall more than 100,000 weights was used to test this algorithm. The application analyzed is recognizing handwritten character images. A database of 800 digits was used for training and 2000 handwritten digits were used for testing. A description of this network can be found in (Le Cun et al., 1990). Figure 2 shows the learning curves on the test set for various unit functions and discretization processes.

First, it should be noted that the results given by the sigmoid function and the saturated ramp with full precision on unit values, gradients, and weights are very similar. This is actually a well known behavior. The surprising result comes from the fact that reducing the precision of the unit values and the gradients to a 1 bit mantissa does not reduce the classification accuracy and does not even slow down the learning process. During these tests the learning process was interrupted at various stages to check that both the unit values (including the input layer, but excluding the output layer) and the gradients (all gradients) were restricted to powers of two. It was further confirmed that only 2 bits were sufficient for the exponent of the unit

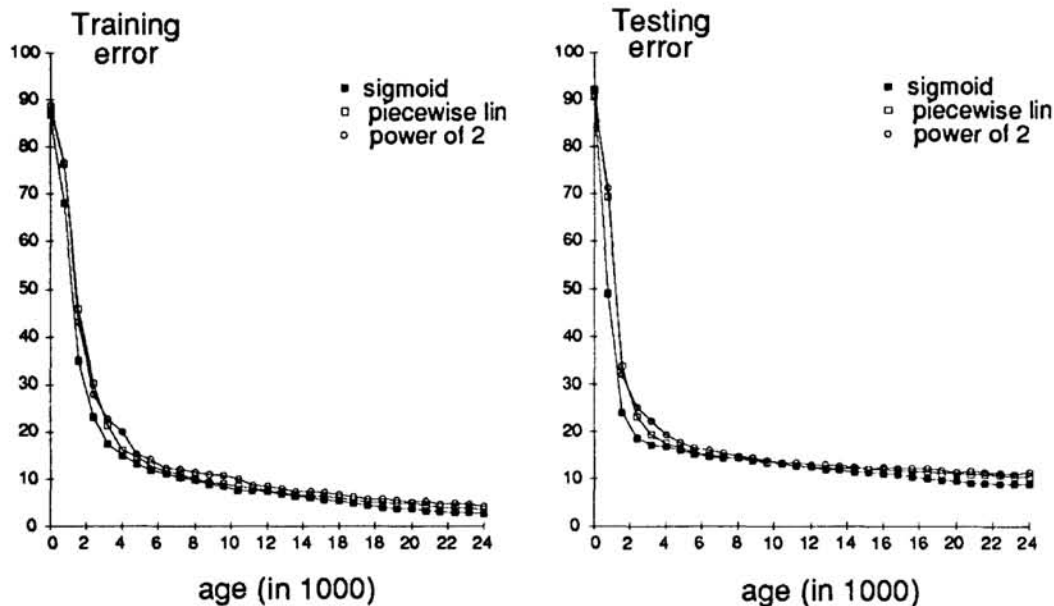

Figure 2: Training and testing error during learning. The filled squares (resp. empty squared) represent the points obtained with the vanilla backpropagation and a sigmoid function (resp. piecewise linear function) used as an activation function. The circles represent the same experiment done with a power of two function used as the activation function, and with all unit gradients discretized to the nearest power of two.

values (from $2^0$ to $2^{-3}$) and 4 bits were sufficient for the exponent of the gradients (from $2^0$ to $2^{-15}$).

To test whether there was any asymptotic limit on performance, we ran a long term experiment (several days) with our largest network (17,000 free parameters) for handwritten character recognition. The training set (60,000 patterns) was made out 30,000 patterns of the original NIST training set (easy) and 30,000 patterns out of the original NIST testing set (hard). Using the most basic backpropagation algorithm (with a guessed constant learning rate) we got the training raw error rate down to under 1% in 20 epochs which is comparable to our standard learning time. Performance on the test set was not as good with the discrete network (it took twice as long to reach equal performance with the discrete network). This was attributed to the unnecessary discretization of the output units [1].

These results show that gradients and unit activations can be discretized to powers of two with negligible loss in performance and convergence speed! The next section will present theoretical explanations for why this is at all possible and why it is generally the case.

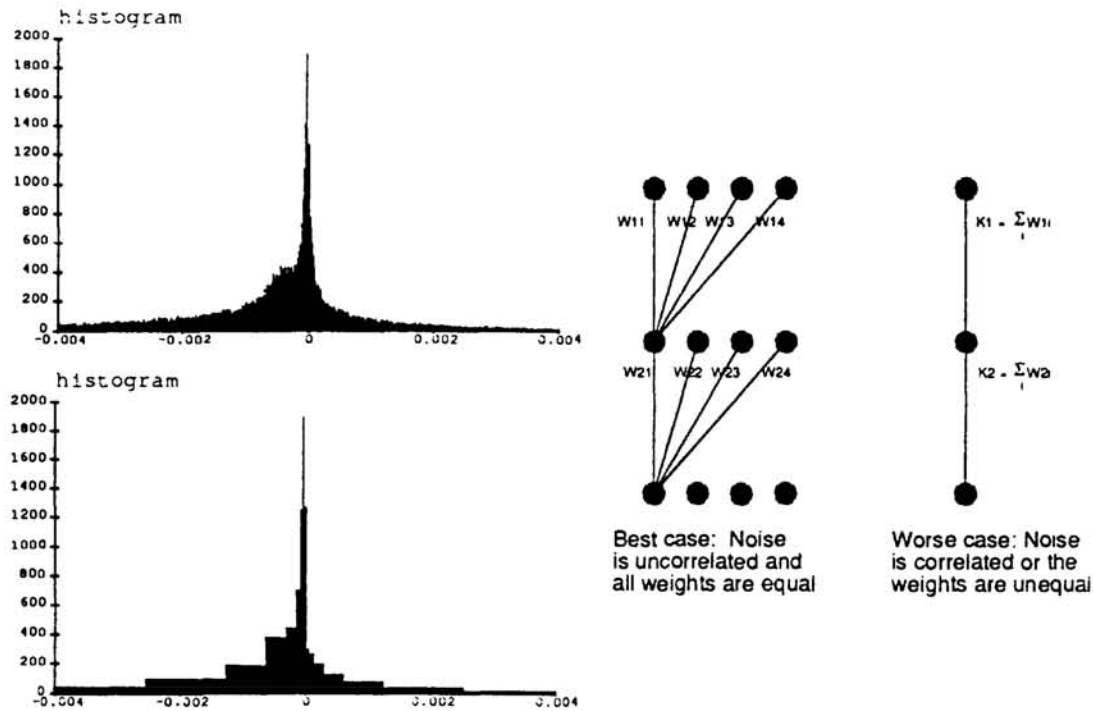

Figure 3: Top left: histogram of the gradients of one output unit after more than 20 epochs of learning over a training set of 60.000 patterns. Bottom left: same histogram assuming that the distribution is constant between powers of two. Right: simplified network architectures for noise effect analysis.

## 4   DISCUSSION

Discretizing the gradient is potentially very dangerous. Convergence may no longer be guaranteed, learning may become prohibitively slow, and final performance after learning may be be too poor to be interesting. We will now explain why these problems do not arise for our choice of discretization.

Let $g_i(p)$ be the error gradient at weight $i$ and pattern $p$. Let $\mu_i$ and $\sigma_i$ be the mean and standard deviation of $g_i(p)$ over the set of patterns. The mean $\mu_i$ is what is driving the weights to their final values, the standard deviation $\sigma_i$ represents the amplitudes of the variations of the gradients from pattern to pattern. In batch learning, only $\mu_i$ is used for the weight update, while in stochastic gradient descent, each $g_i(p)$ is used for the weight update. If the learning rate is small enough the effects of the noise (measured by $\sigma_i$) of the stochastic variable $\sigma_i(p)$ are negligible, but the frequent weight updates in stochastic gradient descent result in important speedups.

To explain why the discretization of the gradient to a power of two has negligible effect on the performance, consider that in stochastic gradient descent, the noise on the gradient is already so large that it is minimally affected by a rounding (of the gradient) to the nearest power of two. Indeed asymptotically, the gradient average ($\mu_i$) tends to be negligible compared to its standard deviation ($\sigma_i$), meaning that from pattern to pattern the gradient can undergo sign reversals. Rounding to the nearest power of two in comparison is a change of at most 33%, but never a change in sign. This additional noise can therefore be compensated by a slight decrease in the learning rate which will hardly affect the learning process.

The histogram of $g_i(p)$ after learning in the last experiment described in the result section, is shown in Figure 3 (over the training set of 60,000 patterns). It is easy to see in the figure that $\mu_i$ is small with respect to $\sigma_i$ (in this experiment $\mu_i$ was one to two orders of magnitude smaller than $\sigma_i$ depending on the layer). We can also see that rounding each gradient to the nearest power of two will not affect significantly the variance $\sigma_i$ and therefore the learning rate will not need to be decreased to achieve the same performance.

We will now try to evaluate the rounding to the nearest power of two effect more quantitatively. The standard deviation of the gradient for any weight can be written as

$$\sigma_i^2 = \frac{1}{N}\sum_p (g_i(p) - \mu_i)^2 = \frac{1}{N}\sum_p g_i(p)^2 - \mu^2 \approx \frac{1}{N}\sum_p g_i(p)^2 \qquad (4)$$

This approximation is very good asymptotically (after a few epochs of learning). For instance if $|\mu_i| < \sigma_i/10$, the above formula gives the standard deviation to 1%.

Rounding the gradient $g_i$ to the nearest power of two (while keeping the sign) can be viewed as the effect of a multiplicative noise $n_i$ in the equation

$$g_i' = 2^k = g_i(1 + n_i) \qquad \text{for some } k \qquad (5)$$

where $g_i'$ is the nearest power of two from $g_i$. It can be easily verified that this implies that $n_i$ ranges from $-1/3$ and $1/3$. From now on, we will view $n_i$ as a random variable which models as noise the effect of discretization. To simplify the computation we will assume that $n_i$ has uniform distribution. The effect of this assumption is depicted in figure 3, where the bottom histogram has been assumed constant between any two powers of two.

To evaluate the effect of the noise $n_i$ in a multilayer network, let $n_{li}$ be the multiplicative noise introduced at layer $l$ ($l = 1$ for the output, and $l = L$ for the first layer above the input) for weight $i$. Let's further assume that there is only one unit per layer (a simplified diagram of the network architecture is shown on figure 3. This is the worst case analysis. If there are several units per layer, the gradients will be summed to units in a lower layer. The gradients within a layer are correlated from unit to unit (they all originate from the same desired values), but the noise introduced by the discretization can only be less correlated, not more. The summation of the gradient in a lower layer can therefore only decrease the effect of the discretization. The worst case analysis is therefore when there is only one unit per layer as depicted in figure 3, extreme right. We will further assume that the noise introduced by the discretization in one layer is independent from the noise introduced in the next layer. This is not really true but it greatly simplifies the derivation.

Let $\mu_i'$ and $\sigma_i'$ be the mean and standard deviation of $g_i(p)'$. Since $n_{li}$ has a zero mean, $\mu_i' = \mu_i$ and $\mu_i'$ is negligible with respect to $g_i(p)$. In the worst case, when the gradient has to be backpropagated all the way to the input, the standard deviation is:

$$\sigma'^2 = \frac{1}{N}\sum_p \frac{3^L}{2^L} \underbrace{\int_{-1/3}^{1/3} \cdots \int_{-1/3}^{1/3}}_{L} \left( g_i(p) \prod_l (1 - n_{li}) - \mu' \right)^2 \prod_l dn_{li}$$

$$= \frac{1}{N}\sum_p g_i(p)^2 \prod_l \left( \frac{3}{2}\int_{-1/3}^{1/3}(1 + n_{li})^2 dn_{li} \right) - \mu^2 \approx \sigma_i^2\left(1 + \frac{1}{27}\right)^L \qquad (6)$$

As learning progresses, the minimum average distance of each weight to the weight corresponding to a local minimum becomes proportional to the variance of the noise on that weight, divided by the learning rate. Therefore, asymptotically (which is where most of the time is spent), for a given convergence speed, the learning rate should be inversely proportional to the variance of the noise in the gradient. This means that to compensate the effect of the discretization, the learning rate should be divided by

$$\frac{\sigma_i'}{\sigma_i} = \left( \sqrt{1 + \frac{1}{27}} \right)^L \approx 1.02^L \tag{7}$$

Even for a 10 layer network this value is only 1.2, ($\sigma_i'$ is 20 % larger than $\sigma_i$). The assumption that the noise is independent from layer to layer tends to slightly underestimate this number while the assumption that the noise from unit to unit in the same layer is completely correlated tends to overestimate it.

All things considered, we do not expect that the learning rate should be decreased by more than 10 to 20% for any practical application. In all our simulations it was actually left unchanged!

## 5   HARDWARE

This algorithm is well suited for integrating a large network on a single chip. The weights are implemented with a resolution of 16 bits, while the states need only 1 bit in the mantissa and 3 bits in the exponent, the gradient 1 bit in the mantissa and 5 bits in the exponent, and for the learning rate 1 bits mantissa and 4 bits exponent suffice. In this way, all the multiplications of weights with states, and of gradients with learning rates and states, reduce to add operations of the exponents.

For the forward pass the weights are multiplied with the states and then summed. The multiplication is executed as a shift operation of the weight values. For summing two products their mantissae have to be aligned, again a shift operation, and then they can be added. The partial sums are kept at full resolution until the end of the summing process. This is necessary to avoid losing the influence of many small products. Once the sum is computed, it is then quantized simply by checking the most significant bit in the mantissa. For the backward propagation the computation runs in the same way, except that now the gradient is propagated through the net, and the learning rate has to be taken into account.

The only operations required for this algorithm are 'shift' and 'add'. An ALU implementing these operations with the resolution mentioned can be built with less than 1,000 transistors. In order to execute a network fast, its weights have to be stored on-chip. Otherwise, the time required to transfer weights from external memory onto the chip boundary makes the high compute power all but useless. If storage is provided for 10,000 weights plus 2,000 states, this requires less than 256 kbit of memory. Together with 256 ALUs and circuitry for routing the data, this leads to a circuit with about 1.7 million transistors, where over 80% of them are contained in the memory. This assumes that the memory is implemented with static cells, if dynamic memory is used instead the transistor count drops considerably. . An operating speed of 40 MHz results in a compute rate of 10 billion operations per second. With such a chip a network may be trained at a speed of more than 1 billion weight updates per second.

This algorithm has been optimized for an implementation on a chip, but it can also provide a considerable speed up when executed on a standard computer. Due to the small resolution of the numbers, several states can be packed into a 32 bit number and hence many more fit into a chache. Moreover on a machine without a hardware multiplier, where the multiplication is executed with microcode, shift operations may be much faster than multiplications. Hence a substancial speedup may be observed.

## Footnotes

[1]Since the output units are not multiplied by anything, there is no need to use a discrete activation function. As a matter of fact the continuous sigmoid function can be implemented by just changing the target values (using the inverse sigmoid function) and by using no activation function for the output units. This modification was not introduced but we believe it would improves the performance on the test set, especially when fancy decision rules (with confidence evaluation) are used, since they require high precision on the output units.

## References

Asanovic, K., Morgan, N., and Wawrzynek, J. (1993). Using Simulations of Reduced Precision Arithmetic to Design a Neuro- Microprocessor. *J. VLSI Signal Processing*, 6(1):33–44.

Jabri, M. and Flower, B. (1992). Weight Perturbation : An optimal architecture and learning technique for analog VLSI feedforward and recurrent multilayer networks. *IEEE Trans. Neural Networks*, 3(3):154–157.

Kwan, H. and Tang, C. (1993). Multipyerless Multilayer Feedforward Neural Network Design Suitable for Continuous Input-Output Mapping. *Electronic Letters*, 29(14):1259–1260.

Le Cun, Y., Boser, B., Denker, J. S., Henderson, D., Howard, R. E., Hubbard, W., and Jackel, L. D. (1990). Handwritten Digit Recognition with a Back-Propagation Network. In Touretzky, D., editor, *Neural Information Processing Systems*, volume 2, (Denver, 1989). Morgan Kaufman.

Marchesi, M., Orlando, G., Piazza, F., and Uncini, A. (1993). Fast Neural Networks without Multipliers. *IEEE Trans. Neural Networks*, 4(1):53–62.

Reyneri, L. and Filippi, E. (1991). An analysis on the Performance of Silicon Implementations of Backpropagation Algorithms for Artificial Neural Networks. *IEEE Trans. Computers*, 40(12):1380–1389.

Sakaue, S., Kohda, T., Yamamoto, H., Maruno, S., and Shimeki, Y. (1993). Reduction of Required Precision Bits for Back-Propagation Applied to Pattern Recognition. *IEEE Trans. Neural Networks*, 4(2):270–275.

Vincent, J. and Myers, D. (1992). Weight dithering and Wordlength Selection for Digital Backpropagation Networks. *BT Technology J.*, 10(3):124–133.

White, B. and Elmasry, M. (1992). The Digi-Neocognitron: A Digital Neocognitron Neural Network Model for VLSI. *IEEE Trans. Neural Networks*, 3(1):73–85.
